# Correlation and Interpolation Networks for Real-time Expression Analysis/Synthesis.

**Trevor Darrell, Irfan Essa, Alex Pentland**
Perceptual Computing Group
MIT Media Lab

## Abstract

We describe a framework for real-time tracking of facial expressions that uses neurally-inspired correlation and interpolation methods. A distributed view-based representation is used to characterize facial state, and is computed using a replicated correlation network. The ensemble response of the set of view correlation scores is input to a network based interpolation method, which maps perceptual state to motor control states for a simulated 3-D face model. Activation levels of the motor state correspond to muscle activations in an anatomically derived model. By integrating fast and robust 2-D processing with 3-D models, we obtain a system that is able to quickly track and interpret complex facial motions in real-time.

## 1  INTRODUCTION

An important task for natural and artificial vision systems is the analysis and interpretation of faces. To be useful in interactive systems and in other settings where the information conveyed is of a time critical nature, analysis of facial expressions must occur quickly, or be of little value. However, many of the traditional computer vision methods for estimating and modeling facial state have proved difficult to perform fast enough for interactive settings. We have therefore investigated neurally inspired mechanisms for the analysis of facial expressions. We use neurally plausible distributed pattern recognition mechanisms to make fast and robust assessments of facial state, and multi-dimensional interpolation networks to connect these measurements to a facial model.

There are many potential applications of a system for facial expression analysis. Person-

alized interfaces which sense a users emotional state, ultra-low bitrate video conferencing which sends only facial muscle activations, as well as the enhanced recognition systems mentioned above. We have focused on a application in computer graphics which stresses both the analysis and synthesis components of our system: interactive facial animation.

In the next sections we develop a computational framework for neurally plausible expression analysis, and the connection to a physically-based face model using a radial basis function method. Finally we will show the results of these methods applied to the interactive animation task, in which an computer graphics model of a face is rendered in real time, and matches the state of the users face as sensed through a conventional video camera.

## 2    EXPRESSION MODELING/TRACKING

The modeling and tracking of expressions and faces has been a topic of increasing interest recently. In the neural network field, several successful models of character expression modeling have been developed by Poggio and colleagues. These models apply multi-dimensional interpolation techniques, using the radial basis function method, to the task of interpolating 2D images of different facial expression. Librande [4] and Poggio and Brunelli [9] applied the Radial Basis Function (RBF) method to facial expression modeling, using a line drawing representation of cartoon faces. In this model a small set of canonical expressions is defined, and intermediate expressions constructed via the interpolation technique. The representation used is a generic "feature vector", which in the case of cartoon faces consists of the contour endpoints. Recently, Beymer et al. [1] extended this approach to use real images, relying on optical flow and image warping techniques to solve the correspondence and prediction problems, respectively.

RBF-based techniques have the advantage of allowing for the efficient and fast computation of intermediate states in a representation. Since the representation is simple and the interpolation computation straight-forward, real-time implementations are practical on conventional systems. These methods interpolate between a set of 2D views, so the need for an explicit 3-D representation is sidestepped. For many applications, this is not a problem, and may even be desirable since it allows the extrapolation to "impossible" figures or expressions, which may be of creative value. However, for realistic rendering and recognition tasks, the use of a 3-D model may be desirable since it can detect such impossible states.

In the field of computer graphics, much work has been done on on the 3-D modeling of faces and facial expression. These models focus on the geometric and physical qualities of facial structure. Platt and Badler [7], Pieper [6], Waters [11] and others have developed models of facial structure, skin dynamics, and muscle connections, respectively, based on available anatomical data. These models provide strong constraints for the tracking of feature locations on a face. Williams et. al. [12] developed a method in which explicit feature marks are tracked on a 3-D face by use of two cameras. Terzopoulos and Waters [10] developed a similar method to track linear facial features, estimate corresponding parameters of a three dimensional wireframe face model, and reproduce facial expression. A significant limitation of these systems is that successful tracking requires facial markings. Essa and Pentland [3] applied optical flow methods (see also Mase [5]) for the passive tracking of facial motion, and integrated the flow measurement method into a dynamic system model. Their method allowed for completely passive estimation of facial expressions, using all the constraints provided by a full 3-D model of facial expression.

Both the view based method of Beymer et. al. and the 3-D model of Essa and Pentland rely

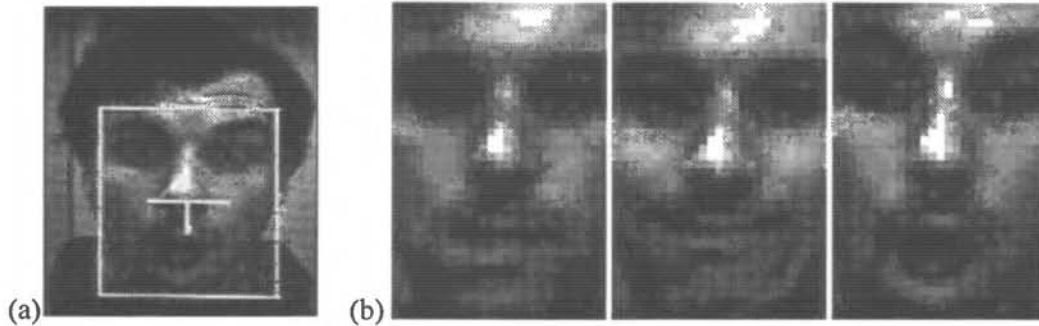

Figure 1: *(a) Frame of video being processed to extract view model. Outlined rectangle indicates area of image used for model. (b) View models found via clustering method on training sequence consisting of neutral, smile, and surprise expressions.*

on estimates of optic flow, which are difficult to compute reliably, especially in real-time. Our approach here is to combine interpolated view-based measurements with physically based models, to take advantage of the fast interpolation capability of the RBF and the powerful constraints imposed by physically based models. We construct a framework in which perceptual states are estimated from real video sequences and are interpolated to control the motor control states of a physically based face model.

## 3   VIEW-BASED FACE PERCEPTION

To make reliable real-time measurements of a complex dynamic object, we use a distributed representation corresponding to distinct views of that object. Previously, we demonstrated the use of this type of representation for the tracking and recognition of hand gestures [2]. Like faces, hands are complex objects with both non-rigid and rigid dynamics. Direct use of a 3-D model for recognition has proved difficult for such objects, so we developed a view-based method for representation. Here we apply this technique to the problem of facial representation, but extend the scheme to connect to a 3-D model for high-level modeling and generation/animation. With this, we gain the representational power and constraints implied by the 3-D model as a high-level representation; however the 3-D model is only indirectly involved in the perception stage, so we can still have the same speed and reliability afforded by the view-based representation.

In our method each view characterizes a particular aspect or pose of the object being represented. The view is stored iconically, that is, it is a literal image or template (but with some point-wise statistics) of the appearance of the object in that aspect or pose. A match criteria is defined between views and input images; usually a normalized correlation function is used, but other criteria are possible. An input image is represented by the ensemble of match scores from that image to the stored views.

To achieve invariance across a range of transformations, for example translation, rotation and/or scale, units which compute the match score for each view are replicated at different values of each transformation.[1] The unit which has maximal response across all values of the transformation is selected, and the ensemble response of the view units which share the

same transformation values as the selected unit is stored as the representation for the input image. We set the perceptual state **X** to be a vector containing this ensemble response.

If the object to be represented is fully known a priori, then methods to generate views can be constructed by analysis of the aspect graph if the object is polyhedral, or in general by rendering images of the object at evenly spaced rotations. However, in practice good 3-D models that are useful for describing image intensity values are rare[2], so we look to data-driven methods of acquiring object views.

As described in [2] a simple clustering algorithm can find a set of views that "span" a training sequence of images, in the sense that for each image in the sequence at least one view is within some threshold similarity to that image. The algorithm is as follows. Let $V$ be the current set of views for an object (initially one view is specified manually). For each frame $I$ of a training sequence, if at least one $v \in V$ has a match value $M(v, I)$ that is greater than a threshold $\theta$, then no action is performed and the next frame is processed. If no view is close, then $I$ is used to construct a new view which is added to the view set. A view $v'$ is created using a window of $I$ centered at the location in the previous image where the closest view was located. (All views usually share the same window size, determined by the initial view.) The view set is then augmented to include the new view: $V = V \cup v'$.

This algorithm will find a set of views which well-characterizes an object across the range of poses or expressions contained in the training sequence. For example, in the domain of hand gestures, inputing a training sequence consisting of a waving hand will yield views which contain images of the hand at several different rotations. In the domain of faces, when input a training sequence consisting of a user performing 3 different expressions, neutral, smile, and surprise, this algorithm (with normalized correlation and $\theta = 0.7$) found three views corresponding to these expressions to represent the face, as shown in Figure 1(b). These 3 views serve as a good representation for the face of this user as long as his expression is similar to one in the training set.

The major advantage of this type of distributed view-based representation lies in the reduction of the dimensionality of the processing that needs to occur for recognition, tracking, or control tasks. In the gesture recognition domain, this dimensionality reduction allowed for conventional recognition strategies to be applied successfully and in real-time, on examples where it would have been infeasible to evaluate the recognition criteria on the full signal. In the domain explored in this paper it makes the interpolation problem of much lower order: rather than interpolate from thousands of input dimensions as would be required when the input is the image domain, the view domain for expression modeling tasks typically has on the order of a dozen dimensions.

## 4   3-D MODELING/MOTOR CONTROL

To model the structure of the face and the dynamics of expression performance, we use the physically based model of Essa et. al. This model captures how expressions are generated by muscle actuations and the resulting skin and tissue deformations. The model is capable of controlled nonrigid deformations of various facial regions, in a fashion similar to how humans generate facial expressions by muscle actuations attached to facial tissue. Finite Element methods are used to model the dynamics of the system.

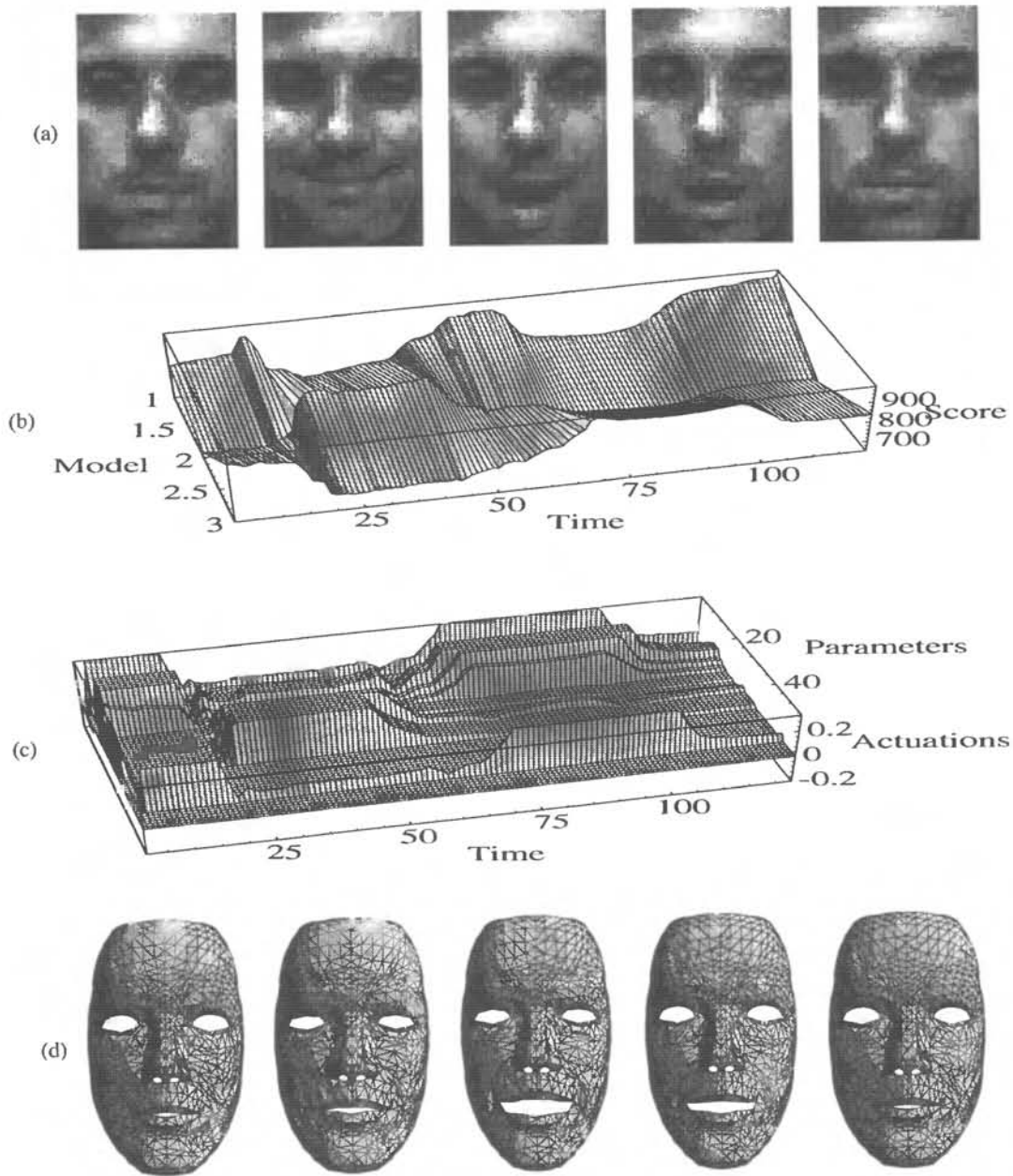

Figure 2: *(a) Face images used as input, (b) normalized correlation scores* $\mathbf{X}(t)$ *for each view model, (c) resulting muscle control parameters* $\mathbf{Y}(t)$*, (d) rendered images of facial model corresponding to muscle parameters.*

This model is based on the mesh developed by Platt and Badler [7], extended into a topologically invariant physics-based model through the addition of a dynamic skin and muscle model [6, 11]. These methods give the facial model an *anatomically-based* facial structure by modeling facial tissue/skin, and muscle actuators, with a geometric model to describe force-based deformations and control parameters.

The muscle model provides us with a set of control knobs to drive the facial state, defined to be a vector $\mathbf{Y}$. These serve to define the motor state of the animated face. Our task now is to connect the perceptual states of the observed face to these motor states.

## 5   CONNECTING PERCEPTION WITH ACTION

We need to establish a mapping from the perceptual view scores to the appropriate muscle activations on the 3-D face model. To do this, we use multidimensional interpolation strategies implemented in network form.

Interpolation requires a set of control points or exemplars from which to derive the desired mapping. Example pairs of real faces and model faces for different expressions are presented to the interpolation method during a training phase. This can be done in one of two ways, with either a user-driven or model-driven paradigm. In the model-driven case the muscle states are set to generate a particular expression by an animator/programmer and then the user is asked to make the equivalent expression. The resulting perceptual (view-model) scores are then recorded and paired with the muscle activation levels. In the user-driven case, the user makes an expression of his/her own choosing, and the optic flow method of Essa et. al. is used to derive the corresponding muscle activation levels. The model-driven paradigm is simpler and faster, but the user-driven paradigm yields more detailed and authentic facial expressions.

We use the Radial Basis Function (RBF) method presented in [8], and define the interpolated motor controls to be a weighted sum of radial functions centered at each example:

$$\mathbf{Y} = \sum_{i=1}^{n} c_i \mathcal{G}(\mathbf{X} - \mathbf{X_i}) \tag{1}$$

where $\mathbf{Y}$ are the muscle states, $\mathbf{X}$ are the observed view-model scores, $\mathbf{X_i}$ are the example scores, $\mathcal{G}$ is an RBF (and in our case was simply a linear ramp $\mathcal{G}(\S) = ||\S||$), and the weights $c_i$ are computed from the example motor values $\mathbf{Y_i}$ using the pseudo-inverse method [8].

## 6   INTERACTIVE ANIMATION SYSTEM

The correlation network, RBF interpolator, and facial model described above have been combined into a single system for interactive animation. The entire system can be updated at over 5 Hz, using a dedicated single board accelerator to compute the correlation network, and an SGI workstation to render the facial mesh. Here we present two examples of the processing performed by the system, using different strategies for coupling perceptual and motor state.

Figure 2 illustrates one example of real-time facial expression tracking using this system, using a full-coupling paradigm. Across the top, labeled (a), are five frames of a video sequence of a user making a smile expression. This was one of the expressions used in the training sequence for the view models shown in Figure 1(b), so they were applicable to be

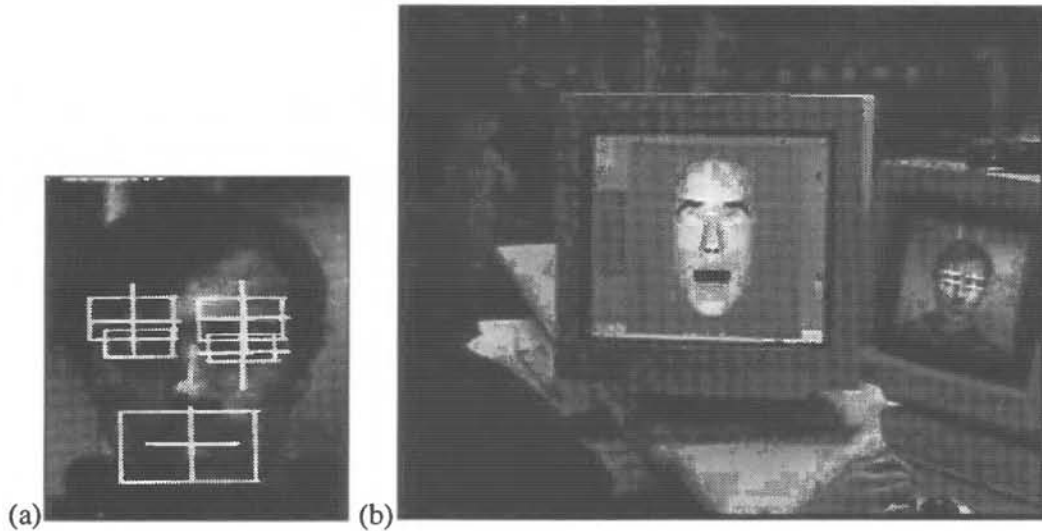

Figure 3: *(a) Processing of video frame with independent view model regions for eyes, eye-brows, and mouth region. (b) Overview shot of full system. User is on left, vision system and camera is on right, and animated face is in the center of the scene. The animated face matches the state of the users face in real-time, including eye-blinks (as is the case in this shot.)*

used here. Figure 2(b) shows the correlation scores computed for each of the 3 view models for each frame of the sequence. This constituted the perceptual state representation, $\mathbf{X}(t)$.

In this example the full face is coupled with the full suite of motor control parameters. An RBF interpolator was trained using perceptual/motor state pairs for three example full-face expressions (neutral, smile, surprise); the resulting (interpolated) motor control values, $\mathbf{Y}(t)$, for the entire sequence are shown in Figure 2(c). Finally, the rendered facial mesh for five frames of these motor control values is shown in Figure 2(d).

When there are only a few canonical expressions that need be tracked/matched, this full-face template approach is robust and simple. However if the user wishes to exercise independent control of the various regions of the face, then the full coupling paradigm will be overly restrictive. For example, if the user trains two expressions, eyes closed and eyes open, and then runs the system and attempts to blink only one eye, the rendered face will be unable to match it. (In fact closing one eye leads to the rendered face half-closing both eyes.)

A solution to this is to decouple the regions of the face which are independent geometrically (and to some degree, in terms of muscle effect.) Under this paradigm, separate correlation networks are computed for each facial regions, and multiple RBF interpolations are performed for each system. Each interpolator drives a distinct subset of the motor state vector. Figure 3(a) shows the regions used for decoupled local templates. In these examples independent regions were used for each eye, eyebrow, and the mouth region.

Finally, figure 3 (b) shows a picture of the set-up of the system as it is being run in an interactive setting. The animated face mimics the facial state of the user, matching in real time the position of the eyes, eyelids, eyebrows and mouth of the user. In the example shown in this picture, the users eyes are closed, so the animated face's eyes are similarly closed. Realistic performance of animated facial expressions and gestures are are possible

through this method, since the timing and levels of the muscle activations react immediately to changes in the users face.

## 7   CONCLUSION

We have explored the use of correlation networks and Radial Basis Function techniques for the tracking of real faces in video sequences. A distributed view-based representation is computed using a network of replicated normalized correlation units, and offers a fast and robust assesment of perceptual state. 3-D constraints on facial shape are achieved through the use of a an anatomically derived facial model, whose muscle activations are controled via interolated perceptual states using the RBF method.

With this framework we have been able to acheive the fast and robust analysis and synthesis of facial expressions. A modeled face mimics the expression of a user in real-time, using only a conventional video camera sensor and no special marking on the face of the user. This system has promise as a new approach in the interactive animation, video tele-conferencing, and personalized interface domains.

## Footnotes

[1]In a computer implementation this exhaustive sampling may be impractical due to the number of units needed, in which case this stage may be approximated by methods which are hybrid sampling/search methods.

[2]As opposed to modeling forces and shape deformations, for which 3-D models are useful and indeed are used in the method presented here.

## References

[1] D. Beymer, A. Shashua, and T. Poggio, Example Based Image Analysis and Synthesis, MIT AI Lab TR-1431, 1993.

[2] T. Darrell and A. Pentland. Classification of Hand Gestures using a View-Based Distributed Representation In *NIPS-6*, 1993.

[3] I. Essa and A. Pentland. A vision system for observing and extracting facial action parameters. In *Proc. IEEE Conf. Computer Vision and Pattern Recognition*, 1994.

[4] S. Librande. Example-based Character Drawing. S.M. Thesis, Media Arts and Science/Media Lab, MIT. 1992

[5] K. Mase. Recognition of facial expressions for optical flow. *IEICE Transactions, Special Issue on Computer Vision and its Applications*, E 74(10), 1991.

[6] S. Pieper, J. Rosen, and D. Zeltzer. Interactive graphics for plastic surgery: A task level analysis and implementation. *Proc. Siggraph-92*, pages 127–134, 1992.

[7] S. M. Platt and N. I. Badler. Animating facial expression. *ACM SIGGRAPH Conference Proceedings*, 15(3):245–252, 1981.

[8] T. Poggio and F. Girosi. A theory of networks for approximation and learning. MIT AI Lab TR-1140, 1989.

[9] T. Poggio and R. Brunelli, A Novel Approach to Graphics, MIT AI Lab TR- 1354. 1992.

[10] D. Terzopoulus and K. Waters. Analysis and synthesis of facial image sequences using physical and anatomical models. *IEEE Trans. PAMI*, 15(6):569–579, June 1993.

[11] K. Waters and D. Terzopoulos. Modeling and animating faces using scanned data. *The Journal of Visualization and Computer Animation*, 2:123–128, 1991.

[12] L. Williams. Performance-driven facial animation. *ACM SIGGRAPH Conference Proceedings*, 24(4):235–242, 1990.
